# IMPLICATIONS OF
# RECURSIVE DISTRIBUTED REPRESENTATIONS

Jordan B. Pollack
Laboratory for AI Research
Ohio State University
Columbus, OH 43210

## ABSTRACT

I will describe my recent results on the automatic development of fixed-width recursive distributed representations of variable-sized hierarchal data structures. One implication of this work is that certain types of AI-style data-structures can now be represented in fixed-width analog vectors. Simple inferences can be performed using the type of pattern associations that neural networks excel at. Another implication arises from noting that these representations become self-similar in the limit. Once this door to chaos is opened, many interesting new questions about the representational basis of intelligence emerge, and can (and will) be discussed.

## INTRODUCTION

A major problem for any cognitive system is the capacity for, and the induction of the potentially infinite structures implicated in faculties such as human language and memory.

Classical cognitive architectures handle this problem through finite but recursive sets of rules, such as formal grammars (Chomsky, 1957). Connectionist architectures, while yielding intriguing insights into fault-tolerance and machine learning, have, thus far, not handled such productive systems in an adequate fashion.

So, it is not surprising that one of the main attacks on connectionism, especially on its application to language processing models, has been on the adequacy of such systems to deal with apparently rule-based behaviors (Pinker & Prince, 1988) and systematicity (Fodor & Pylyshyn, 1988).

I had earlier discussed precisely these challenges for connectionism, calling them the generative capacity problem for language, and the representational adequacy problem for data structures (Pollack, 1987b). These problems are actually intimately related, as the capacity to recognize or generate novel language relies on the ability to represent the underlying concept.

Recently, I have developed an approach to the representation problem, at least for recursive structures like sequences and trees. Recursive auto-associative memory (RAAM) (Pollack, 1988a). automatically develops recursive distributed representations of finite training sets of such structures, using Back-Propagation (Rumelhart et al., 1986). These representations appear to occupy a novel position in the space of both classical and connectionist symbolic representations.

A fixed-width representation of variable-sized symbolic trees leads immediately to the implication that simple forms of neural-network associative memories may be able to perform inferences of a type that are thought to require complex machinery such as variable binding and unification.

But when we take seriously the *infinite* part of the representational adequacy problem, we are lead into a strange intellectual area, to which the second part of this paper is addressed.

# BACKGROUND

## RECURSIVE AUTO-ASSOCIATIVE MEMORY

A RAAM is composed of two mechanisms: a compressor, and a reconstructor, which are simultaneously trained. The job of the compressor is to encode a small set of fixed-width patterns into a single pattern of the same width. This compression can be recursively applied, from the bottom up, to a fixed-valence tree with distinguished labeled terminals (leaves), resulting in a fixed-width pattern representing the entire structure. The job of the reconstructor is to accurately decode this pattern into its parts, and then to further decode the parts as necessary, until the terminal patterns are found, resulting in a reconstruction of the original tree.

For *binary* trees with $k$-bit binary patterns as the leaves, the compressor could be a single-layer feedforward network with $2k$ inputs and $k$ outputs, along with additional control machinery. The reconstructor could be a single-layer feedforward-network with $k$ inputs and $2k$ outputs, along with a mechanism for testing whether a pattern is a terminal.

We simultaneously train these two networks in an auto-associative framework as follows. Consider the tree, ((D (A N))(V (P (D N)))), as one member of a training set of such trees, where the lexical categories are pre-encoded as k-bit vectors. If the $2k-k-2k$ network is successfully trained (defined below) with the following patterns (among other such patterns in the training environment), the resultant compressor and reconstructor can reliably form representations for these binary trees.

| input pattern | | hidden pattern | | output pattern |
|---|---|---|---|---|
| A+N | $\rightarrow$ | $R_{AN}(t)$ | $\rightarrow$ | A$\prime$+N$\prime$ |
| D+$R_{AN}(t)$ | $\rightarrow$ | $R_{DAN}(t)$ | $\rightarrow$ | D$\prime$+$R_{AN}(t)\prime$ |
| D+N | $\rightarrow$ | $R_{DN}(t)$ | $\rightarrow$ | D$\prime$+N$\prime$ |
| P+$R_{DN}(t)$ | $\rightarrow$ | $R_{PDN}(t)$ | $\rightarrow$ | P$\prime$+$R_{DN}(t)\prime$ |
| V+$R_{PDN}(t)$ | $\rightarrow$ | $R_{VPDN}(t)$ | $\rightarrow$ | V$\prime$+$R_{PDN}(t)\prime$ |
| $R_{DAN}(t)$+$R_{VPDN}(t)$ | $\rightarrow$ | $R_{DANVPDN}(t)$ | $\rightarrow$ | $R_{DAN}(t)\prime$+$R_{VPDN}(t)\prime$ |

The (initially random) values of the hidden units, $R_i(t)$, are part of the training environment, so it (and the representations) evolve along with the weights.[1]

Because the training regime involves multiple compressions, but only single reconstructions, we rely on an induction that the reconstructor works. If a reconstructed pattern, say $R_{PDN}\prime$, is sufficiently close to the original pattern, then its parts can be reconstructed as well.

## AN EXPERIMENT

The tree considered above was one member of the first experiment done on RAAM's. I used a simple context-free parser to parse a set of lexical-category sequences into a set of bracketed binary trees:

$$(D (A (A (A N))))$$
$$((D N)(P (D N)))$$
$$(V (D N))$$
$$(P (D (A N)))$$
$$((D N) V)$$

$$((D N) (V (D (A N)))))$$
$$((D (A N)) (V (P (D N)))))$$

Each terminal pattern (D A N V & P) was represented as a 1-bit-in-5 code padded with 5 zeros. A 20-10-20 RAAM devised the representations shown in figure 1.

Figure 1.

*Representations of all the binary trees in the training set, devised by a 20-10-20 RAAM, manually clustered by phrase-type. The squares represent values between 0 and 1 by area.*

I labeled each tree and its representation by the phrase type in the grammar, and sorted them by type. The RAAM, without having any intrinsic concepts of phrase-type, has clearly developed a representation with similarity between members of the same type. For example, the third feature seems to be clearly distinguishing sentences from non-sentences, the fifth feature seems to be involved in separating adjective phrases from others, while the tenth feature appears to distinguish prepositional and noun phrases from others.[2]

At the same time, the representation must be keeping enough information about the sub-trees in order to allow the reconstructor to accurately recover the original structure. So, knowledge about structural regularity flows into the weights while constraints about context similarity guide the development of the representations.

## RECURSIVE DISTRIBUTED REPRESENTATIONS

These vectors are a very new kind of representation, a **recursive, distributed representation**, hinted at by Hinton's (1988) notion of a reduced description.

They combine aspects of several disparate representations. Like feature-vectors, they are fixed-width, similarity-based, and their content is easily accessible. Like symbols, they combine only in syntactically well-formed ways. Like symbol-structures, they have constituency and compositionality. And, like pointers, they refer to larger symbol structures

which can be efficiently retrieved.

But, unlike feature-vectors, they compose. Unlike symbols, they can be compared. Unlike symbol structures, they are fixed in size. And, unlike pointers, they have content.

Recursive distributed representations could, potentially, lead to a reintegration of syntax and semantics at a very low level[3]. Rather than having meaning-free symbols which syntactically combine, and meanings which are recursively ascribed, we could functionally compose symbols which bear their own meanings.

## IMPLICATIONS

One of the reasons for the historical split between symbolic AI and fields such as pattern recognition or neural networks is that the structured representations AI requires do not easily commingle with the representations offered by n-dimensional vectors.

Since recursive distributed representations form a bridge from structured representations to n-dimensional vectors, they will allow high-level AI tasks to be accomplished with neural networks.

## ASSOCIATIVE INFERENCE

There are many kinds of inferences which seem to be very easy for humans to perform. In fact, we must perform incredibly long chains of inferences in the act of understanding natural language (Birnbaum, 1986).

And yet, when we consider performing those inferences using standard techniques which involve variable binding and unification, the costs seem prohibitive. For humans, however, these inferences seem to cost no more than simple associative priming (Meyer & Schvaneveldt, 1971).

Since RAAMS can devise representations of trees as analog patterns which can actually be associated, they may lead to very fast neuro-logical inference engines.

For example, in a larger experiment, which was reported in (Pollack, 1988b), a 48-16-48 RAAM developed representations for a set of ternary trees, such as

(THOUGHT PAT (KNEW JOHN (LOVED MARY JOHN)))

which corresponded to a set of sentences with complex constituent structure. This RAAM was able to represent, as points within a 16-dimensional hypercube, all cases of (LOVED X Y) where X and Y were chosen from the set {JOHN, MARY, PAT, MAN}.

A simple test of whether or not associative inference were possible, then, would be to build a "symmetric love" network, which would perform the simple inference: "If (LOVED X Y) then (LOVED Y X)".

A network with 16 input and output units and 8 hidden units was successfully trained on 12 of the 16 possible associations, and worked perfectly on the remaining 4. (Note that it accomplished this task without any explicit machinery for matching and moving X and Y.)

One might think that in order to chain simple inferences like this one we will need many hidden layers. But there has recently been some coincidental work showing that feed-

forward networks with two layers of hidden units can compute arbitrary mappings (Lapedes & Farber, 1988a; Lippman, 1987). Therefore, we can assume that the sequential application of associative-style inferences can be speeded up, at least by retraining, to a simple 3-cycle process.

## OPENING THE DOOR TO CHAOS

### The Capacity of RAAM's

As discussed in the introduction, the question of infinite generative capacity is central. In the domain of RAAM's the question becomes: Given a finite set of trees to represent, how can the system then represent an infinite number of related trees.

For the syntactic-tree experiment reported above, the 20-10-20 RAAM was only able to represent 32 new trees. The 48-16-48 RAAM was able to represent many more than it was trained on, but not yet an infinite number in the linguistics sense.

I do not yet have any closed analytical forms for the capacity of a recursive auto-associative memory. Given that is is not really a file-cabinet or content-addressable memory, but a memory for a gestalt of rules for recursive pattern compression and reconstruction, capacity results such as those of (Willshaw, 1981) and (Hopfield, 1982) do not directly apply. Binary patterns are not being stored, so one cannot simply count how many.

I have considered, however, the capacity of such a memory in the limit, where the actual functions and analog representations are not bounded by single linear transformations and sigmoids or by 32-bit floating point resolution.

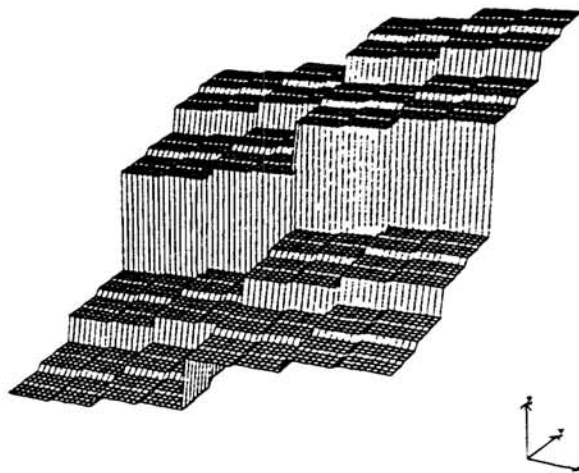

Figure 2.

*A plot of the bit-interspersal function. The x and y axis represent the left and right subtrees, and the height represents the output of the function.*

Consider just a 2-1-2 recursive auto-associator. It is really a reconstructible mapping from points in the unit square to points on the unit line. In order to work, the function should define a parametric 1-dimensional curve in 2-space, perhaps a set of connected splines.[4] As more and more data points need to be encoded, this parametric curve will become more convoluted to cover them. In the limit, it will no longer be a 1-dimensional curve, but a space-filling curve with a fractional dimension.

One possible functional basis for this ultimate 2-1-2 recursive auto-associator is "bit-interspersal," where the compression function would return a number, between 0 and 1, by interleaving the bits of the binary-fractional representations of the left and right sub-trees. Figure 2 depicts this function, not as a space-filling curve, but as a surface, where no two points project to the same height. The surface is a 3-dimensional variant of a recognizable instance of Cantor dust called the devil's staircase.

Thus, it is my working hypothesis that alternative activation functions (i.e. other than the usual sigmoidal or threshold), based on fractal or chaotic mathematics, is the critical missing link between neural networks and infinite capacity systems.

## Between AI and Chaos

The remainder of this paper is what is behind the door; the result of simultaneous consideration of the fields of AI, Neural Networks, Fractals, and Chaos.[5] It is, in essence, a proposal on where (I am planning) to look for fruitful interplay between these fields, and what some interdisciplinary problems are which could be solved in this context.

There has already been some intrusion of interest in chaos in the physics-based study of neural networks as dynamical systems. For example both (Huberman & Hogg, 1987) and (Kurten, 1987) show how phase-transitions occur in particular neural-like systems, and (Lapedes & Farber, 1988b) demonstrate how a network trained to predict a simple iterated function would follow that function's bifurcations into chaos.

However, these efforts are either noticing chaos, or working with it as a domain. At the other end of the spectrum are those relying on chaos to explain such things as the emergence of consciousness, or free will (Braitenberg, 1984, p. 65).

In between these extremes lies some very hard problems recognized by AI which, I believe, could benefit from a new viewpoint.

### Self-Similarity and the Symbol-Grounding Problem

The bifurcation between structure and form which leads to the near universality of discrete symbolic structures with ascribed meanings has lead to a yawning gap between cognitive and perceptual subareas of AI.

This gulf can be seen between such fields as speech recognition and language comprehension, early versus late vision, and robotics versus planning. The low-level tasks require numeric, sensory representations, while the high-level ones require compositional symbolic representations.[6]

The idea of infinitely regressing symbolic representations which bottom-out at perception has been an unimplementable folk idea ("Turtles all the way down") in AI for quite some time.

The reason for its lack of luster is that the amount of information in such a structure is considered combinatorially explosive. Unless, of course, one considers self-similarity to be an information-limiting construction.

---

a complete 2-1-2 RAAM could be found, it would give a unique number to every binary tree such that the number of a tree would be a invertible function of the numbers of its two subtrees.

[5] Talking about 4 disciples is both difficult, and dangerous, considering the current size of the chasm, and the mutual hostilities: AI thinks NN is just a spectre, NN thinks AI is dead, F thinks it subsumes C, and C thinks F is its just showbiz.

[6] It is no surprise then, that neural networks are much more successful at the former tasks.

While working on a new activation function for RAAMS which would magically have this property, I have started building modular systems of RAAMs, following Ballard's (1987) work on non-recursive auto-associators.

When viewing a RAAM as a constrained system, one can see that the terminal patterns are overconstrained and the highest-level non-terminal patterns are unconstrained. Only those non-terminals which are further compressed have a reasonable similarity constraint. One could imagine a cascade of RAAMs, where the highest non-terminal patterns of a low-level RAAM (say, for encodings of letters) are the terminal patterns for a middle-level RAAM (say, for words), whose non-terminal patterns are the terminals for a higher-level RAAM (say, for sentences).

If all the representations were the same width, then there must be natural similarities between the structures at different conceptual scales.

### Induction Inference and Strange Automata

The problem of inductive inference[7], of developing a machine which can learn to recognize or generate a language is a pretty hard problem, even for regular languages.

In the process of extending my work on a recurrent high-order neural network called sequential cascaded nets (Pollack, 1987a), something strange occurred.

It is always possible to completely map out any unknown finite-state machine by providing each known state with every input token, and keeping track of the states. This is, in fact, what defines such a machine as finite.

Since a recurrent network is a dynamical system, rather than an automaton, one must choose a fuzz-factor for comparing real numbers. For a particular network trained on a context-free grammar, I was unable to map it out. Each time I reduced the fuzz-factor, the machine doubled in size, much like Mandelbrot's coastline (Mandelbrot, 1982)

This suggests a bidirectional analogy between finite state automata and dynamical systems of the neural network sort[8]. An automaton has an initial state, a set of states, a lexicon, and and a function which produces a new state given an old state and input token. A subset of states are distinguished as accepting states. A dynamical system has an initial state, and an equation which defines its evolution over time, perhaps in response to environment.

Such dynamical systems have elements known as attractor states, to which the state of the system usually evolves. Two such varieties, limit points and limit cycles, correspond directly to similar elements in finite-state automata, states with loops back to themselves, and short boring cycles of states (such as the familiar "Please Login. Enter Password. Bad Password. Please Login...").

But there is an element in non-linear dynamical systems which does not have a correlate in formal automata theory, which is the notion of a chaotic, or strange, attractor, first noticed in work on weather prediction (Lorenz, 1963). A chaotic attractor does not repeat.

The implications for inductive inference is that while, formally, push-down automata and Turing machines are necessary for recognizing harder classes of languages, such as context-free or context-sensitive, respectively, the idiosyncratic state-table and external memory of such devices make them impossible to induce. On the other hand, chaotic dynamical systems look much like automata, and should be about as hard to induce. The

infinite memory is internal to the state vector, and the finite-state-control is built into a more regular, but non-linear, function.

## Fractal Energy Landscapes and Natural Kinds

Hopfield (1982) described an associative memory in which each of a finite set of binary vectors to be stored would define a local minima in some energy landscape. The Boltzmann Machine (Ackley et al., 1985) uses a similar physical analogy along with simulated annealing to seek the global minimum in such landscapes as well. Pineda (1987) has a continuous version of such a memory, where the attractor states are analog vectors.

One can think of these energy minimization process as a ball rolling down hills. Given a smooth landscape, that ball will roll into a local minima. On the other hand, if the landscape were constructed by recursive similarity, or by a midpoint displacement technique, such as those used in figures of fractal mountains, there will be an infinite number of local minima, which will be detected based on the size of the ball. Naillon and Theeten's report (this volume), in which an exponential number of attractors are used, is along the proposed line.

The idea of high-dimensional feature vectors has a long history in psychological studies of memory and representation, and is known to be inadequate from that perspective as well as from the representational requirements of AI. But AI has no good empirical candidates for a theory of mental representation either.

Such theories generally break down when dealing with novel instances of Natural Kinds, such as birds, chairs, and games. A robot with necessary and sufficient conditions, logical rules, or circumscribed regions in feature space cannot deal with walking into a room, recognizing and sitting on a hand-shaped chair.

If the chairs we know form the large-scale local minima of an associative memory, then perhaps the chairs we don't know can also be found as local minima in the same space, albeit on a smaller scale. Of course, all the chairs we know are only smaller-scale minima in our memory for furniture.

## Fractal Compression and the Capacity of Memory

Consider something like the Mandelbrot set as the basis for a reconstructive memory. Rather than storing all pictures, one merely has to store the "pointer" to a picture,[9] and, with the help of a simple function and large computer, the picture can be retrieved. Most everyone has seen glossy pictures of the colorful prototype shapes of yeasts and dragons that infinitely appear as the location and scale are changed along the chaotic boundary.

The first step in this hypothetical construction is to develop a related set with the additional property that it can be inverted in the following sense: Given a rough sketch of a picture likely to be in the set, return the best "pointer" to it.[10]

The second step, perhaps using normal neural-network technology, is to build an invertible non-linear mapping from the prototypes in a application domain (like chess positions, human faces, sentences, schemata, etc..) to the largest-scale prototypes in the mathematical memory space.

Taken together, this hypothetical system turns out to be a look-up table for an infinite set of similar representations which incurs no memory cost for its contents. Only the pointers and the reconstruction function need to be stored. Such a basis for reconstructive storage would render meaningless the recent attempts at "counting the bits" of human memory (Hillis, 1988; Landauer, 1986).

While these two steps together sound quite fantastic, it is closely related to the RAAM idea using a chaotic activation function. The reconstructor produces contents from pointers, while the compressor returns pointers from contents. And the idea of a uniform fractal basis for memory is not really too distant from the idea of a uniform basis for visual images, such as iterated fractal surfaces based on the collage theorem (Barnsley et al., 1985).

A moral could be that impressive demonstrations of compression, such as the bidirectional mapping from ideas to language, must be easy when one can discover the underlying regularity.

# CONCLUSION

Recursive auto-associative memory can develop fixed-width recursive distributed representations for variable-sized data-structures such as symbolic trees. Given such representations, one implication is that complex inferences, which seemed to require complex information handling strategies, can be accomplished with associations.

A second implication is that the representations must become self-similar and space-filling in the limit. This implication, of fractal and chaotic structures in mental representations, may lead to a reconsideration of many fundamental decisions in computational cognitive science.

Dissonance for cognitive scientists can be induced by comparing the infinite output of a formal language generator (with anybody's rules), to the boundary areas of the Mandelbrot set with its simple underlying function. Which is vaster? Which more natural?

For when one considers the relative success of fractal versus euclidean geometry at compactly describing natural objects, such as trees and coastlines, one must wonder at the accuracy of the pervasive description of naturally-occurring mental objects as features or propositions which bottom-out at meaningless terms.

## Footnotes

[1] This "moving target" strategy is also used by (Elman, 1988) and (Dyer et al., 1988).

[2] In fact, by these metrics, the test case ((D N)(P (D N))) should really be classified as a sentence; since it was not used in any other construction, there was no reason for the RAAM to believe otherwise.

[3] The *wrong distinction* is the inverse of the *undifferentiated concept* problem in science, such as the fusing of the notions of heat and temperature in the 17th century (Wiser & Carey, 1983). For example, a company which manufactured workstations based on a hardware distinction between characters and graphics had deep trouble when trying to build a modern window system...

[4] (Saund, 1987) originally made the connection between auto-association and *dimensionality reduction*. If such

[7] For a good survey see (Angluin & Smith, 1983). J. Feldman recently posed this as a "challenge" problem for neural networks (c.f. Servan-Scrieber, Cleermans, & McClelland (this volume)).

[8] Wolfram (1984) has, of course, made the analogy between dynamical systems and cellular automata.

[9] I.e. a point on the complex plane and the window size

[10] Related sets might show up with great frequency using iterated systems, like Newton's method or back-propagation. And a more precise notion of inversion, involving both representational tolerance and scale, is required.

## References

Ackley, D. H., Hinton, G. E. & Sejnowski, T. J. (1985). A learning algorithm for Boltzmann Machines. *Cognitive Science, 9*, 147-169.

Angluin, D. & Smith, C. H. (1983). Inductive Inference: Theory and Methods. *Computing Surveys, 15*, 237-269.

Ballard, D. H. (1987). Modular Learning in Neural Networks. In *Proceedings of the Sixth National Conference on Artificial Intelligence*. Seattle, 279-284.

Barnsley, M. F., Ervin, V., Hardin, D. & Lancaster, J. (1985). Solution of an inverse problem for fractals and other sets. *Proceedings of the National Academy of Science, 83*.

Birnbaum, L. (1986). Integrated processing in planning and understanding. Research Report 489, New Haven: Computer Science Dept., Yale Univeristy.

Braitenberg, V. (1984). *Vehicles: Experiments in synthetic psychology*. Cambridge: MIT press.

Chomsky, N. (1957). *Syntactic structures*. The Hague: Mouton and Co..

Dyer, M. G., Flowers, M. & Wang, Y. A. (1988). Weight Matrix = Pattern of Activation: Encoding Semantic Networks as Distributed Representations in DUAL, a PDP architecture. UCLA-Artificial Intelligence-88-5, Los Angeles: Artificial Intelligence Laboratory, UCLA.

Elman, J. L. (1988). Finding Structure in Time. Report 8801, San Diego: Center for Research in Language, UCSD.

Fodor, J. & Pylyshyn, A. (1988). Connectionism and Cognitive Architecture: A Critical Analysis. *Cognition, 28*, 3-71.

Hillis, W. D. (1988). Intelligence as emergent behavior; or, the songs of eden. *Daedelus, 117*, 175-190.

Hinton, G. (1988). Representing Part-Whole hierarchies in connectionist networks. In *Proceedings of the Tenth Annual Conference of the Cognitive Science Society*. Montreal, 48-54.

Hopfield, J. J. (1982). Neural Networks and physical systems with emergent collective computational abilities. *Proceedings of the National Academy of Sciences USA, 79*, 2554-2558.

Huberman, B. A. & Hogg, T. (1987). Phase Transitions in Artificial Intelligence Systems. *Artificial Intelligence, 33*, 155-172.

Kurten, K. E. (1987). Phase transitions in quasirandom neural networks. In *Institute of Electrical and Electronics Engineers First International Conference on Neural Networks*. San Diego, II-197-20.

Landauer, T. K. (1986). How much do people remember? Some estimates on the quantity of learned information in long-term memory.. *Cognitive Science, 10*, 477-494.

Lapedes, A. S. & Farber, R. M. (1988). How Neural Nets Work. LAUR-88-418: Los Alamos.

Lapedes, A. S. & Farber, R. M. (1988). Nonlinear Signal Processing using Neural Networks: Prediction and system modeling. *Biological Cybernetics, To appear.*

Lippman, R. P. (1987). An introduction to computing with neural networks. *Institute of Electrical and Electronics Engineers ASSP Magazine, April,* 4-22.

Lorenz, E. N. (1963). Deterministic Nonperiodic Flow. *Journal of Atmospheric Sciences, 20*, 130-141.

Mandelbrot, B. (1982). *The Fractal Geometry of Nature*. San Francisco: Freeman.

Meyer, D. E. & Schvaneveldt, R. W. (1971). Facilitation in recognizing pairs of words: Evidence of a dependence between retrieval operations. *Journal of Experimental Psychology, 90*, 227-234.

Pineda, F. J. (1987). Generalization of Back-Propagation to Recurrent Neural Networks. *Physical Review Letters, 59*, 2229-2232.

Pinker, S. & Prince, A. (1988). On Language and Connectionism: Analysis of a parallel distributed processing model of language inquisition.. *Cognition, 28*, 73-193.

Pollack, J. B. (1987). Cascaded Back Propagation on Dynamic Connectionist Networks. In *Proceedings of the Ninth Conference of the Cognitive Science Society*. Seattle, 391-404.

Pollack, J. B. (1987). On Connectionist Models of Natural Language Processing. Ph.D. Thesis, Urbana: Computer Science Department, University of Illinois. (Available as MCCS-87-100, Computing Research Laboratory, Las Cruces, NM)

Pollack, J. B. (1988). Recursive Auto-Associative Memory: Devising Compositional Distributed Representations. In *Proceedings of the Tenth Annual Conference of the Cognitive Science Society*. Montreal, 33-39.

Pollack, J. B. (1988). Recursive Auto-Associative Memory: Devising Compositional Distributed Representations. MCCS-88-124, Las Cruces: Computing Research Laboratory, New Mexico State University.

Rumelhart, D. E., Hinton, G. & Williams, R. (1986). Learning Internal Representations through Error Propagation. In D. E. Rumelhart, J. L. McClelland & the PDP research Group, (Eds.), *Parallel Distributed Processing: Experiments in the Microstructure of Cognition*, Vol. 1. Cambridge: MIT Press.

Saund, E. (1987). Dimensionality Reduction and Constraint in Later Vision. In *Proceedings of the Ninth Annual Conference of the Cognitive Science Society*. Seattle, 908-915.

Willshaw, D. J. (1981). Holography, Associative Memory, and Inductive Generalization. In G. E. Hinton & J. A. Anderson, (Eds.), *Parallel models of associative memory*. Hillsdale: Lawrence Erlbaum Associates.

Wiser, M. & Carey, S. (1983). When heat and temperature were one. In D. Gentner & A. Stevens, (Eds.), *Mental Models*. Hillsdale: Erlbaum.

Wolfram, S. (1984). Universality and Complexity in Cellular Automata. *Physica, 10D*, 1-35.
